# On the optimality of incremental neural network algorithms

**Ron Meir***
Department of Electrical Engineering
Technion, Haifa 32000, Israel
rmeir@dumbo.technion.ac.il

**Vitaly Maiorov[†]**
Department of Mathematics
Technion, Haifa 32000, Israel
maiorov@tx.technion.ac.il

## Abstract

We study the approximation of functions by two-layer feedforward neural networks, focusing on incremental algorithms which greedily add units, estimating single unit parameters at each stage. As opposed to standard algorithms for fixed architectures, the optimization at each stage is performed over a small number of parameters, mitigating many of the difficult numerical problems inherent in high-dimensional non-linear optimization. We establish upper bounds on the error incurred by the algorithm, when approximating functions from the Sobolev class, thereby extending previous results which only provided rates of convergence for functions in certain convex hulls of functional spaces. By comparing our results to recently derived lower bounds, we show that the greedy algorithms are nearly optimal. Combined with estimation error results for greedy algorithms, a strong case can be made for this type of approach.

## 1 Introduction and background

A major problem in the application of neural networks to real world problems is the excessively long time required for training large networks of a fixed architecture. Moreover, theoretical results establish the intractability of such training in the worst case [9][4]. Additionally, the problem of determining the architecture and size of the network required to solve a certain task is left open. Due to these problems, several authors have considered incremental algorithms for constructing the network by the addition of hidden units, and estimation of each unit's parameters incrementally. These approaches possess two desirable attributes: first, the optimization is done step-wise, so that only a small number of parameters need to be optimized at each stage; and second, the structure of the network

*This work was supported in part by the a grant from the Israel Science Foundation

†The author was partially supported by the center for Absorption in Science, Ministry of Immigrant Absorption, State of Israel.

is established concomitantly with the learning, rather than specifying it in advance. However, until recently these algorithms have been rather heuristic in nature, as no guaranteed performance bounds had been established. Note that while there has been a recent surge of interest in these types of algorithms, they in fact date back to work done in the early seventies (see [3] for a historical survey).

The first theoretical result establishing performance bounds for incremental approximations in Hilbert space, was given by Jones [8]. This work was later extended by Barron [2], and applied to neural network approximation of functions characterized by certain conditions on their Fourier coefficients. The work of Barron has been extended in two main directions. First, Lee *et al.* [10] have considered approximating general functions using Hilbert space techniques, while Donahue *et al.* [7] have provided powerful extensions of Jones' and Barron's results to general Banach spaces. One of the most impressive results of the latter work is the demonstration that iterative algorithms can, in many cases, achieve nearly optimal rates of convergence, when approximating convex hulls.

While this paper is concerned mainly with issues of approximation, we comment that it is highly relevant to the statistical problem of learning from data in neural networks. First, Lee *et al.* [10] give estimation error bounds for algorithms performing incremental optimization with respect to the training error. Under certain regularity conditions, they are able to achieve rates of convergence comparable to those obtained by the much more computationally demanding algorithm of empirical error minimization. Moreover, it is well known that upper bounds on the approximation error are needed in order to obtain performance bounds, both for parametric and nonparametric estimation, where the latter is achieved using the method of complexity regularization. Finally, as pointed out by Donahue *et al.* [7], lower bounds on the approximation error are crucial in establishing worst case speed limitations for learning.

The main contribution of this paper is as follows. For functions belonging to the Sobolev class (see definition below), we establish, under appropriate conditions, near-optimal rates of convergence for the incremental approach, and obtain explicit bounds on the parameter values of the network. The latter bounds are often crucial for establishing estimation error rates. In contrast to the work in [10] and [7], we characterize approximation rates for functions belonging to standard smoothness classes, such as the Sobolev class. The former work establishes rates of convergence with respect to the convex hulls of certain subsets of functions, which do not relate in a any simple way to standard functional classes (such as Lipschitz, Sobolev, Hölder, etc.). As far as we are aware, the results reported here are the first to report on such bounds for incremental neural network procedures. A detailed version of this work, complete with the detailed proofs, is available in [13].

## 2   Problem statement

We make use of the nomenclature and definitions from [7]. Let $\mathcal{H}$ be a Banach space of functions with norm $\| \cdot \|$. For concreteness we assume henceforth that the norm is given by the $L_q$ norm, $1 < q < \infty$, denoted by $\| \cdot \|_q$. Let $\mathrm{lin}_n \mathcal{H}$ consist of all sums of the form $\sum_{i=1}^n a_i g_i$, $g_i \in \mathcal{H}$ and arbitrary $a_i$, and $\mathrm{co}_n \mathcal{H}$ is the set of such sums with $a_i \in [0, 1]$ and $\sum_{i=1}^n a_i = 1$. The distances, measured in the $L_q$ norm, from a function $f$ are given by

$$\mathrm{dist}(\mathrm{lin}_n \mathcal{H}, f) = \inf \left\{ \|h - f\|_q : h \in \mathrm{lin}_n \mathcal{H} \right\},$$
$$\mathrm{dist}(\mathrm{co}_n \mathcal{H}, f) = \inf \left\{ \|h - f\|_q : h \in \mathrm{co}_n \mathcal{H} \right\}.$$

The *linear span* of $\mathcal{H}$ is given by $\mathrm{lin}\mathcal{H} = \cup_n \mathrm{lin}_n \mathcal{H}$, while the *convex-hull* of $\mathcal{H}$ is $\mathrm{co}\mathcal{H} = \cup_n \mathrm{co}_n \mathcal{H}$. We follow standard notation and denote closures of sets by a bar, e.g. $\overline{\mathrm{co}\mathcal{H}}$ is the closure of the convex hull of $\mathcal{H}$. In this work we focus on the special case where

$$\mathcal{H} = \mathcal{H}_\eta \stackrel{\triangle}{=} \left\{ g : g(x) = c\sigma(a^T x + b), |c| \le \eta, \|\sigma(\cdot)\|_q \le 1 \right\}, \qquad (1)$$

corresponding to the basic building blocks of multilayer neural networks. The restriction $\|\sigma(\cdot)\| \leq 1$ is not very demanding as many sigmoidal functions can be expressed as a sum of functions of bounded norm. It should be obvious that $\lin_n \mathcal{H}_\eta$ corresponds to a two-layer neural network with a linear output unit and $\sigma$-activation functions in the single hidden layer, while $\co_n \mathcal{H}_\eta$ is equivalent to a restricted form of such a network, where restrictions are placed on the hidden-to-output weights. In terms of the definitions introduced above, the by now well known property of universal function approximation over compacta can be stated as $\overline{\lin \mathcal{H}} = C(M)$, where $C(M)$ is the class of continuous real valued functions defined over $M$, a compact subset of $\mathbf{R}^d$. A necessary and sufficient condition for this has been established by Leshno *et al.* [11], and essentially requires that $\sigma(\cdot)$ be locally integrable and non-polynomial. We comment that if $\eta = \infty$ in (1), and $c$ is unrestricted in sign, then $\co \mathcal{H}_\infty = \lin \mathcal{H}_\infty$. The distinction becomes important only if $\eta < \infty$, in which case $\co \mathcal{H}_\eta \subset \lin \mathcal{H}_\eta$.

For the purpose of incremental approximation, it turns out to be useful to consider the convex hull $\co \mathcal{H}$, rather than the usual linear span, as powerful algorithms and performance bounds can be developed in this case. In this context several authors have considered bounds for the approximation of a function $f$ belonging to $\overline{\co \mathcal{H}}$ by sequences of functions belonging to $\co_n \mathcal{H}$. However, it is not clear in general how well convex hulls of *bounded* functions approximate general functions. One contribution of this work is to show how one may control the rate of growth of the bound $\eta$ in (1), so that general functions, belonging to certain smoothness classes (e.g. Sobolev), may be well approximated. In fact, we show that the incremental approximation scheme described below achieves nearly optimal approximation error for functions in the Sobolev space.

Following Donahue *et al.* [7], we consider $\varepsilon$-greedy algorithms. Let $\varepsilon = (\varepsilon_1, \varepsilon_2, \dots)$ be a positive sequence, and similarly for $(\alpha_1, \alpha_2, \dots)$, $0 < \alpha_n < 1$. A sequence of functions $h_1, h_2, \dots$ is $\varepsilon$-greedy with respect to $f$ if for $n = 0, 1, 2, \dots$,

$$\|h_{n+1} - f\|_q < \inf \{\|\alpha_n h_n + (1 - \alpha_n)g - f\|_q : g \in \mathcal{H}_\eta\} + \varepsilon_n, \tag{2}$$

where we set $h_0 = 0$. For simplicity we set $\alpha_n = (n-1)/n$, although other schemes are also possible. It should be clear that at each stage $n$, the function $h_n$ belongs to $\co_n \mathcal{H}_\eta$. Observe also that at each step, the infimum is taken with respect to $g \in \mathcal{H}_\eta$, the function $h_n$ being fixed. In terms of neural networks, this implies that the optimization over each hidden unit parameters $(a, b, c)$ is performed independently of the others. We note in passing, that while this greatly facilitates the optimization process in practice, no theoretical guarantee can be made as to the convexity of the single-node error function (see [1] for counter-examples). The variables $\varepsilon_n$ are slack variables, allowing the extra freedom of only approximate minimization. In this paper we do not optimize over $\alpha_n$, but rather fix a sequence in advance, forfeiting some generality at the price of a simpler presentation. In any event, the rates we obtain are unchanged by such a restriction.

In the sequel we consider $\varepsilon$-greedy approximations of smooth functions belonging to the Sobolev class of functions,

$$W_2^r = \left\{ f : \max_{0 \leq \vec{k} \leq r} \|\mathcal{D}^{\vec{k}} f\|_2 \leq 1 \right\},$$

where $\vec{k} = (k_1, \dots, k_d)$, $k_i \geq 0$ and $|\vec{k}| = k_1 + \cdots k_d$. Here $\mathcal{D}^{\vec{k}}$ is the partial derivative operator of order $\vec{k}$. All functions are defined over a compact domain $K \subset \mathbf{R}^d$.

## 3 Upper bound for the $L_2$ norm

First, we consider the approximation of functions from $W_2^r$ using the $L_2$ norm. In distinction with other $L_q$ norms, there exists an inner product in this case, defined through

$(\cdot, \cdot) = \| \cdot \|_2^2$. This simplification is essential to the proof in this case.

We begin by recalling a result from [12], demonstrating that any function in $L_2$ may be exactly expressed as a *convex* integral representation of the form

$$f(x) = Q \int h(x, \theta) w(\theta) d\theta, \qquad (3)$$

where $0 < Q < \infty$ depends on $f$, and $w(\theta)$ is a probability density function (pdf) with respect to the multi-dimensional variable $\theta$. Thus, we may write $f(x) = Q \mathrm{E}_w \{h(x, \Theta)\}$, where $\mathrm{E}_w$ denotes the expectation operator with respect to the pdf $w$. Moreover, it was shown in [12], using the Radon and wavelet transforms, that the function $h(x, \theta)$ can be taken to be a ridge function with $\theta = (a, b, c)$ and $h(x, \theta) = c\sigma(a^T x + b)$.

In the case of neural networks, this type of convex representation was first exploited by Barron in [2], assuming $f$ belongs to a class of functions characterized by certain moment conditions on their Fourier transforms. Later, Delyon *et al.* [6] and Maiorov and Meir [12] extended Barron's results to the case of wavelets and neural networks, respectively, obtaining rates of convergence for functions in the Sobolev class.

The basic idea at this point is to generate an approximation, $h_n(x)$, based on $n$ draws of random variables $\Theta^n = \{\Theta_1, \Theta_2, \ldots, \Theta_n\}$, $\Theta_i \sim w(\cdot)$, resulting in the random function

$$h_n(x; \Theta^n) = \frac{Q}{n} \sum_{i=1}^{n} h(x, \Theta_i). \qquad (4)$$

Throughout the paper we conform to standard notation, and denote random variables by uppercase letters, as in $\Theta$, and their realization by lower case letters, as in $\theta$. Let $w^n = \prod_{i=1}^{n} w_i$ represent the product pdf for $\{\Theta_1, \ldots, \Theta_n\}$. Our first result demonstrates that, on the average, the above procedure leads to good approximation of functions belonging to $W_2^r$.

**Theorem 3.1** *Let $K \subset \mathbf{R}^d$ be a compact set. Then for any $f \in W_2^r$, $n > 0$ and $\varepsilon > 0$ there exists a constant $c > 0$, such that*

$$\mathrm{E}_{w^n} \| f - h_n(x; \Theta^n) \|_2 \leq cn^{-r/d + \varepsilon}, \qquad (5)$$

*where $Q < cn^{(1/2 - r/d)+}$, and $(x)_+ = \max(0, x)$.*

The implication of the upper bound on the expected value, is that there exists a set of values $\theta^{*,n} = \{\theta_1^*, \ldots, \theta_n^*\}$, for which the rate (5) can be achieved. Moreover, as long as the functions $h(x, \theta_i)$ in (4) are bounded in the $L_2$ norm, a bound on $Q$ implies a bound on the size of the function $h_n$ itself.

**Proof sketch**    The proof proceeds by expressing $f$ as the sum of two functions, $f_1$ and $f_2$. The function $f_1$ is the best approximation to $f$ from the class of multi-variate splines of degree $r$. From [12] we know that there exist parameters $\theta^n$ such that $\| f_1(\cdot) - h_n(\cdot, \theta^n) \|_2 \leq cn^{-r/d}$. Moreover, using the results of [5] it can be shown that $\| f_2 \|_2 \leq cn^{-r/d}$. Using these two observations, together with the triangle inequality $\| f - h_n \|_2 \leq \| f_1 - h_n \|_2 + \| f_2 \|_2$, yields the desired result.                    ∎

Next, we show that given the approximation rates attained in Theorem 3.1, the same rates may be obtained using an $\varepsilon$-greedy algorithm. Moreover, since in [12] we have established the optimality of the upper bound (up to a logarithmic factor in $n$), we conclude that greedy approximations can indeed yield near-optimal performance, while at the same time being much more attractive computationally. In fact, in this section we use a weaker algorithm, which does not perform a full minimization at each stage.

**Incremental algorithm:** $(q = 2)$ Let $\alpha_n = 1 - 1/n, \bar{\alpha}_n = 1 - \alpha_n = 1/n$.

    1. Let $\theta_1^*$ be chosen to satisfy

$$\|f(x) - Qh(x, \theta_1^*)\|_2^2 = \mathrm{E}_{w_1} \left\{ \|f(x) - Qh(x, \Theta_1)\|_2^2 \right\}.$$

    2. Assume that $\theta_1^*, \theta_2^*, \dots, \theta_{n-1}^*$ have been generated. Select $\theta_n^*$ to obey

$$\left\| f(x) - \frac{Q\alpha_n}{n-1} \sum_{i=1}^{n-1} h(x, \theta_i^*) - \bar{\alpha}_n Qh(x, \theta_n^*) \right\|_2^2$$

$$= \mathrm{E}_{w_n} \left\{ \left\| f(x) - \frac{Q\alpha_n}{n-1} \sum_{i=1}^{n-1} h(x, \theta_i^*) - \bar{\alpha}_n Qh(x, \Theta_n) \right\|_2^2 \right\}.$$

Define

$$\mathrm{E}_n^{(i)} \triangleq \mathrm{E}_{w_n} \left\{ \left\| f(x) - \frac{Q\alpha_n}{n-1} \sum_{i=1}^{n-1} h(x, \theta_i^*) - \bar{\alpha}_n Qh(x, \Theta_n) \right\|_2^2 \right\}, \quad \cdot$$

which measures the error incurred at the $n$-th stage by this incremental procedure. The main result of this section then follows.

**Theorem 3.2** *For any $f \in W_2^r$ and $\varepsilon > 0$, the error of the incremental algorithm above is bounded as*

$$\mathrm{E}_n^{(i)} \le cn^{-\frac{r}{d}+\varepsilon},$$

*for some finite constant c.*

**Proof sketch** The claim will be established upon showing that

$$\mathrm{E}_n^{(i)} \triangleq \mathrm{E}_{w^n} \left\{ \left\| f(x) - \frac{Q}{n} \sum_{i=1}^{n} h(x, \Theta_i) \right\|_2^2 \right\}, \tag{6}$$

namely, the error incurred by the incremental procedure is identical to that of the non-incremental one described preceding Theorem 3.1. The result will then follow upon using Hölder's inequality and the upper bound (5) for the r.h.s. of (6). The remaining details are straightforward, but tedious, and can be found in the full paper [13]. $\blacksquare$

## 4   Upper bound for general $L_q$ norms

Having established rates of convergence for incremental approximation of $W_2^r$ in the $L_2$ norm, we move now to general $L_q$ norms. First, note that the proof of Theorem 3.2 relies heavily on the existence on an inner product. This useful tool is no longer available in the case of general Banach spaces such as $L_q$. In order to extend the results to the latter norm, we need to use more advanced ideas from the theory of the geometry of Banach spaces. In particular, we will make use of recent results from the work of Donahue et al. [7]. Second, we must keep in mind that the approximation of the Sobolev space $W_2^r$ using the $L_q$ norm only makes sense if the *embedding condition $r/d > (1/2 - 1/q)_+$* holds, since otherwise the $L_q$ norm may be infinite (the embedding condition guarantees its finiteness; see [14] for details).

We first present the main result of this section, followed by a sketch of the proof. The full details of the rather technical proof can be found in [13]. Note that in this case we need to use the greedy algorithm (2) rather than the algorithm of Section 3.

**Theorem 4.1** *Let the embedding condition $r/d > (1/2 - 1/q)_+$ hold for $1 < q < \infty$, $0 < r < r^*$, $r^* = \frac{d}{2} + \left(\frac{1}{2} - \frac{1}{q}\right)_+$ and assume that $\|h(\cdot, \theta)\|_q \leq 1$ for all $\theta$. Then for any $f \in W_2^r$ and $\epsilon > 0$*

$$\|f(\cdot) - h_n(\cdot, \theta^n)\|_q \leq cn^{-\gamma + \varepsilon},$$

*where*

$$\gamma = \begin{cases} \frac{\frac{r}{d} - \left(\frac{1}{2} - \frac{1}{q}\right)}{\frac{1}{d} + \frac{2}{q} - \frac{2}{qd}} & q > 2, \\ \frac{r}{d} & q \leq 2, \end{cases} \tag{7}$$

$c = c(r, d, K)$ *and* $h_n(\cdot, \theta^n)$ *is obtained via the incremental greedy algorithm (2) with* $\varepsilon_n = 0$.

**Proof sketch** The main idea in the proof of Theorem 4.1 is a two-part approximation scheme. First, based on [13], we show that any $f \in W_2^r$ may be well approximated by functions in the convex class $\text{co}_n(\mathcal{H}_\eta)$ for an appropriate value of $\eta$ (see Lemma 5.2 in [13]), where $H_\eta$ is defined in (1). Then, it is argued, making use of results from [7] (in particular, using Corollary 3.6), that an incremental greedy algorithm can be used to approximate the closure of the class $\text{co}(H_\eta)$ by the class $\text{co}_n(\mathcal{H}_\eta)$. The proof is completed by using the triangle inequality. The proof along the above lines is done for the case $q > 2$. In the case $q \leq 2$, a simple use of the Hölder inequality in the form $\|f\|_q \leq |K|^{1/q-1/2}\|f\|_2$, where $|K|$ is the volume of the region $K$, yields the desired result, which, given the lower bounds in [12], is nearly optimal. ∎

## 5 Discussion

We have presented a theoretical analysis of an increasingly popular approach to incremental learning in neural networks. Extending previous results, we have shown that near-optimal rates of convergence may be obtained for approximating functions in the Sobolev class $W_2^r$. These results extend and clarify previous work dealing solely with the approximation of functions belonging to the closure of convex hulls of certain sets of functions. Moreover, we have given explicit bounds on the parameters used in the algorithm, and shown that the restriction to $\text{co}_n \mathcal{H}_\eta$ is not too stringent. In the case $q \leq 2$ the rates obtained are as good (up to logarithmic factors) to the rates obtained for general spline functions, which are known to be optimal for approximating Sobolev spaces. The rates obtained in the case $q > 2$ are sub-optimal as compared to spline functions, but can be shown to be provably better than any *linear* approach. In any event, we have shown that the rates obtained are equal, up to logarithmic factors, to approximation from $\text{lin}_n \mathcal{H}_\eta$, when the size of $\eta$ is chosen appropriately, implying that positive input-to-output weights suffice for approximation. An open problem remaining at this point is to demonstrate whether incremental algorithms for neural network construction can be shown to be optimal for every value of $q$. In fact, this is not even known at this stage for neural network approximation in general.

## References

[1] P. Auer, M. Herbster, and M. Warmuth. Exponentially many local minima for single neurons. In D.S. Touretzky, M.C. Mozer, and M.E. Hasselmo, editors, *Advances in Neural Information Processing Systems 8*, pages 316–322. MIT Press, 1996.

[2] A.R. Barron. Universal approximation bound for superpositions of a sigmoidal function. *IEEE Trans. Inf. Th.*, 39:930–945, 1993.

[3] A.R. Barron and R.L. Barron. Statistical learning networks: a unifying view. In E. Wegman, editor, *Computing Science and Statistics: Proceedings 20th Symposium Interface*, pages 192–203, Washington D.C., 1988. Amer. Statis. Assoc.

[4] A. Blum and R. Rivest. Training a 3-node neural net is np-complete. In D.S. Touretzky, editor, *Advances in Neural Information Processing Systems I*, pages 494–501. Morgan Kaufmann, 1989.

[5] C. de Boor and G. Fix. Spline approximation by quasi-interpolation. *J. Approx. Theory*, 7:19–45, 1973.

[6] B. Delyon, A. Juditsky, and A. Benveniste. Accuracy nalysis for wavelet approximations. *IEEE Transaction on Neural Networks*, 6:332–348, 1995.

[7] M.J. Donahue, L. Gurvits, C. Darken, and E. Sontag. Rates of convex approximation in non-hilbert spaces. *Constructive Approx.*, 13:187–220, 1997.

[8] L. Jones. A simple lemma on greedy approximation in Hilbert space and convergence rate for projection pursuit regression and neural network training. *Ann. Statis.*, 20:608–613, 1992.

[9] S. Judd. *Neural Network Design and the Complexity of Learning*. MIT Press, Boston, USA, 1990.

[10] W.S. Lee, P.S. Bartlett, and R.C. Williamson. Efficient Agnostic learning of neural networks with bounded fan-in. *IEEE Trans. Inf. Theory*, 42(6):2118–2132, 1996.

[11] M. Leshno, V. Lin, A. Pinkus, and S. Schocken. Multilayer Feedforward Networks with a Nonpolynomial Activation Function Can Approximate any Function. *Neural Networks*, 6:861–867, 1993.

[12] V.E. Maiorov and R. Meir. On the near optimality of the stochastic approximation of smooth functions by neural networks. Technical Report CC-223, Technion, Department of Electrical Engineering, November 1997. Submitted to *Advances in Computational Mathematics*.

[13] R. Meir and V. Maiorov. On the optimality of neural network approximation using incremental algorithms. *Submitted for publication*, October 1998. ftp://dumbo.technion.ac.il/pub/PAPERS/incremental.pdf.

[14] H. Triebel. *Theory of Function Spaces*. Birkhauser, Basel, 1983.